# What can a single neuron compute?

**Blaise Agüera y Arcas,**[1] **Adrienne L. Fairhall,**[2] **and William Bialek**[2]
[1]Rare Books Library, Princeton University, Princeton, New Jersey 08544
[2]NEC Research Institute, 4 Independence Way, Princeton, New Jersey 08540
*blaisea@princeton.edu* {*adrienne,bialek*} *@research.nj.nec.com*

## Abstract

In this paper we formulate a description of the computation performed by a neuron as a combination of dimensional reduction and nonlinearity. We implement this description for the Hodgkin-Huxley model, identify the most relevant dimensions and find the nonlinearity. A two dimensional description already captures a significant fraction of the information that spikes carry about dynamic inputs. This description also shows that computation in the Hodgkin-Huxley model is more complex than a simple integrate-and-fire or perceptron model.

## 1 Introduction

Classical neural network models approximate neurons as devices that sum their inputs and generate a nonzero output if the sum exceeds a threshold. From our current state of knowledge in neurobiology it is easy to criticize these models as over-simplified: where is the complex geometry of neurons, or the many different kinds of ion channel, each with its own intricate multistate kinetics? Indeed, progress at this more microscopic level of description has led us to the point where we can write (almost) exact models for the electrical dynamics of neurons, at least on short time scales. These nearly exact models are complicated by any measure, including tens if not hundreds of differential equations to describe the states of different channels in different spatial compartments of the cell. Faced with this detailed microscopic description, we need to answer a question which goes well beyond the biological context: given a continuous dynamical system, what does it compute?

Our goal in this paper is to make this question about what a neuron computes somewhat more precise, and then to explore what we take to be the simplest example, namely the Hodgkin–Huxley model [1],[2] (and refs therein).

## 2 What do we mean by the question?

Real neurons take as inputs signals at their synapses and give as outputs sequences of discrete, identical pulses—action potentials or 'spikes'. The inputs themselves are spikes from other neurons, so the neuron is a device which takes $N \sim 10^3$ pulse trains as inputs and generates one pulse train as output. If the system operates at 2 msec resolution and the window of relevant inputs is 20 msec, then we can think of a single neuron as having an input described by a $\sim \times 10^4$ bit word—the presence or absence of a spike in each 2 msec bin for each presynaptic cell—which is then mapped to a one (spike) or zero (no spike). More realistically, if the average spike

rates are $\sim 10\,\text{sec}^{-1}$, the input words can be compressed by a factor of ten. Thus we might be able to think about neurons as evaluating a Boolean function of roughly 1000 Boolean variables, and then characterizing the computational function of the cell amounts to specifying this Boolean function.

The above estimate, though crude, makes clear that there will be no direct empirical attack on the question of what a neuron computes: there are too many possibilities to learn the function by brute force from any reasonable set of experiments. Progress requires the hypothesis that the function computed by a neuron is not arbitrary, but belongs to a simple class. Our suggestion is that this simple class involves functions that vary only over a low dimensional subspace of the inputs, and in fact we will start by searching for linear subspaces.

Specifically, we begin by simplifying away the spatial structure of neurons and take inputs to be just injected currents into a point–like neuron. While this misses some of the richness in real cells, it allows us to focus on developing our computational methods. Further, it turns out that even this simple problem is not at all trivial. If the input is an injected current, then the neuron maps the history of this current, $I(t < t_0)$, into the presence or absence of a spike at time $t_0$. More generally we might imagine that the cell (or our description) is noisy, so that there is a probability of spiking $P[\text{spike@}t_0|I(t < t_0)]$ which depends on the current history. We emphasize that the dependence on the *history* of the current means that there still are many dimensions to the input signal even though we have collapsed any spatial variations. If we work at time resolution $\Delta t$ and assume that currents in a window of size $T$ are relevant to the decision to spike, then the inputs live in a space of $D = T/\Delta t$, of order 100 dimensions in many interesting cases.

If the neuron is sensitive only to a low dimensional linear subspace, we can define a set of signals $s_1, s_2, \cdots, s_K$ by filtering the current,

$$s_\mu = \int_0^\infty dt\, f_\mu(t) I(t_0 - t), \qquad (1)$$

so that the probability of spiking depends only on this finite set of signals,

$$P[\text{spike@}t_0|I(t < t_0)] = P[\text{spike@}t_0]g(s_1, s_2, \cdots, s_K), \qquad (2)$$

where we include the average probability of spiking so that $g$ is dimensionless. If we think of the current $I(t < t_0)$ as a vector, with one dimension for each time sample, then these filtered signals are linear projections of this vector.

In this formulation, characterizing the computation done by a neuron means estimating the number of relevant stimulus dimensions ($K$, hopefully much less than $D$), identifying the filters which project into this relevant subspace,[1] and then characterizing the nonlinear function $g(\vec{s})$. The classical perceptron–like cell of neural network theory has only one relevant dimension and a simple form for $g$.

## 3 Identifying low–dimensional structure

The idea that neurons might be sensitive only to low–dimensional projections of their inputs was developed explicitly in work on a motion sensitive neuron of the fly visual system [3]. Rather than looking at the distribution $P[\text{spike@}t_0|s(t < t_0)]$, with $s(t)$ the input signal (velocity of motion across the visual field in [3]), that work considered the distribution of signals conditional on the response, $P[s(t < t_0)|\text{spike@}t_0]$; these are related by Bayes' rule,

$$\frac{P[\text{spike@}t_0|s(t < t_0)]}{P[\text{spike@}t_0]} = \frac{P[s(t < t_0)|\text{spike@}t_0]}{P[s(t < t_0)]}. \qquad (3)$$

Within the response conditional ensemble $P[s(t < t_0)|\text{spike@}t_0]$ we can compute various moments. Thus the spike triggered average stimulus, or reverse correlation function [4], is the first moment

$$STA(\tau) = \int [ds] \, P[s(t < t_0)|\text{spike@}t_0]s(t_0 - \tau) \,. \tag{4}$$

We can also compute the covariance matrix of fluctuations around this average,

$$C_{\text{spike}}(\tau, \tau') = \int [ds] \, P[s(t < t_0)|\text{spike@}t_0]s(t_0 - \tau)s(t_0 - \tau') - STA(\tau)STA(\tau'). \tag{5}$$

In the same way that we compare the spike triggered average to some constant average level of the signal (which we can define to be zero) in the whole experiment, we want to compare the covariance matrix $C_{\text{spike}}$ with the covariance of the signal averaged over the whole experiment,

$$C_{\text{prior}}(\tau, \tau') = \int [ds] \, P[s(t < t_0)]s(t_0 - \tau)s(t_0 - \tau'). \tag{6}$$

Notice that all of these covariance matrices are $D \times D$ in size. The surprising finding of [3] was that the change in the covariance matrix, $\Delta C = C_{\text{spike}} - C_{\text{prior}}$, had only a very small number of nonzero eigenvalues. In fact it can be shown that if the probability of spiking depends on $K$ linear projections of the stimulus as in eq. (2), and if the inputs $s(t)$ are chosen from a Gaussian distribution, then the rank of the matrix $\Delta C$ is exactly $K$. Further, the eigenvectors associated with nonzero eigenvalues span the relevant subspace (up to a rotation associated with the auto-correlations in the inputs. Thus eigenvalue analysis of the spike triggered covariance matrix gives us a direct way to search for a low dimensional linear subspace that captures the relevant stimulus features.

## 4 The Hodgkin–Huxley model

We recall the details of the Hodgkin–Huxley model and note some special features that guide our analysis. Hodgkin and Huxley [1] modeled the dynamics of the current through a patch of membrane by flow through ion–specific conductances:

$$I(t) = C\frac{dV}{dt} + \bar{g}_K n^4 (V - V_K) + \bar{g}_{Na} m^3 h (V - V_{Na}) + \bar{g}_l (V - V_l), \tag{7}$$

where $K$ and $Na$ subscripts denote potassium– and sodium–related variables, respectively, and $l$ (for 'leakage') terms are a catch-all for other ion species with slower dynamics. $C$ is the membrane capacitance. The subscripted voltages $V_l$ and $V_{Na}$ are ion-specific reversal potentials. $\bar{g}_l$, $\bar{g}_K$ and $\bar{g}_{Na}$ are empirically determined maximal conductances for the different ions,[2] and the gating variables $n$, $m$ and $h$ (on the interval $[0, 1]$) have their own voltage dependent dynamics:

$$
\begin{aligned}
dn/dt &= (0.01V + 0.1)(1 - n)\exp(-0.1V) - 0.125n\exp(V/80) \\
dm/dt &= (0.1V + 2.5)(1 - m)\exp(-0.1V - 1.5) - 4m\exp(V/18) \\
dh/dt &= 0.07(1 - h)\exp(0.05V) - h\exp(-0.1V - 4),
\end{aligned} \tag{8}
$$

with $V$ in mV and $t$ in msec.

Here we are interested in dynamic inputs $I(t)$, but it is important to remember that for constant inputs the Hodgkin–Huxley model undergoes a Hopf bifurcation to spike at a constant frequency; further, this frequency is rather insensitive to the precise value of the input above onset. This 'rigidity' of the system is felt also in

many regimes of dynamic stimulation, and can be thought of as a strong interaction among successive spikes. These interactions lead to long memory times, reflecting the infinite phase memory of the periodic orbit which exists for constant input. While spike interactions are interesting, we want to focus on the way that input current modulates the probability of spiking. To separate these effects we consider only 'isolated' spikes. These are defined by accumulating the interspike interval distribution and noticing that for some intervals $t > t_c$ the distribution decays exponentially, which means that the system has lost memory of the previous spike; thus spikes which are more than $t_c$ after the previous spike are isolated.

In what follows we consider the response of the Hodgkin–Huxley model to currents $I(t)$ with zero mean, 0.275 nA standard deviation, and 0.5 msec correlation time.

## 5   How many dimensions?

Fig. 1 shows the change in covariance matrix $\Delta C(\tau, \tau')$ for isolated spikes in our HH simulation, and fig. 2(a) shows the resulting spectrum of eigenvalues as a function of sample size. The result strongly suggests that there are many fewer than $D$ relevant dimensions. In particular, there seem to be two outstanding modes; the STA itself lies largely in the subspace of these modes, as shown in Fig. 2(b).

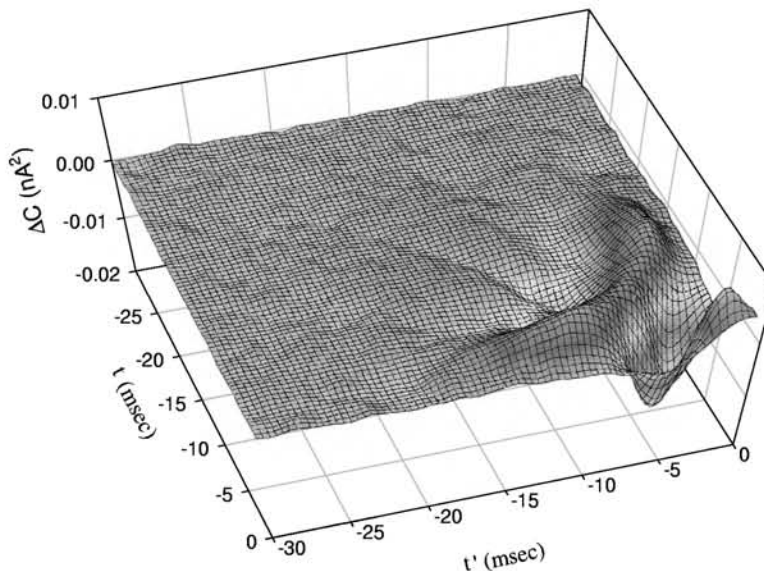

Figure 1: The isolated spike triggered covariance matrix $\Delta C(\tau, \tau')$.

The filters themselves, shown in fig. 3, have simple forms; in particular the second mode is almost exactly the derivative of the first. If the neuron filtered its inputs and generated a spike when the output of the filter crosses threshold, we would find that there are two significant dimensions, corresponding to the filter and its derivative. It is tempting to suggest, then, that this is a good approximation to the HH model, but we will see that this is not correct. Notice also that both filters have significant differentiating components—the cell is not simply integrating its inputs.

Although fig. 2(a) suggests that two modes dominate, it also demonstrates that the smaller nonzero eigenvalues of the other modes are not just noise. The width of any spectral band of eigenvalues near zero due to finite sampling should decline with increasing sample size. However, the smaller eigenvalues seen in fig. 2(a) are stable. Thus while the system is primarily sensitive to two dimensions, there is something

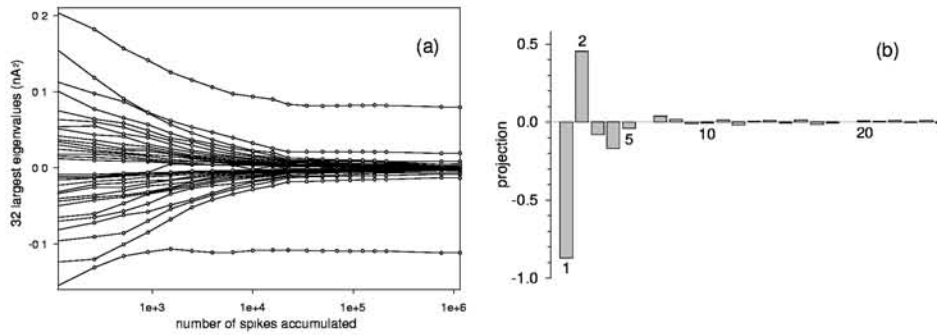

Figure 2: (a) Convergence of the largest 32 eigenvalues of the isolated spike triggered covariance with increasing sample size. (b) Projections of the isolated STA onto the covariance modes.

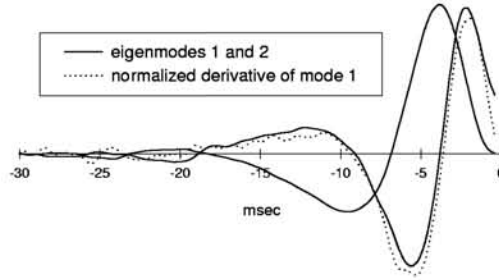

Figure 3: Most significant two modes of the spike-triggered covariance.

missing in this picture. To quantify this, we must first characterize the nonlinear function $g(s_1, s_2)$.

# 6 Nonlinearity and information

At each instant of time we can find the relevant projections of the stimulus $s_1$ and $s_2$. By construction, the distribution of these signals over the whole experiment, $P(s_1, s_2)$, is Gaussian. On the other hand, each time we see a spike we get a sample from the distribution $P(s_1, s_2|\text{spike@}t_0)$, leading to the picture in fig. 4. The prior and spike conditional distributions clearly are better separated in two dimensions than in one, which means that our two dimensional description captures more than the spike triggered average. Further, we see that the spike conditional distribution is curved, unlike what we would expect for a simple thresholding device.

Combining eq's. (2) and (3), we have

$$g(s_1, s_2) = \frac{P(s_1, s_2|\text{spike@}t_0)}{P(s_1, s_2)}, \qquad (9)$$

so that these two distributions determine the input/output relation of the neuron in this 2D space. We emphasize that although the subspace is linear, $g$ can have arbitrary nonlinearity. Fig. 4 shows that this input/output relation has sharp edges, but also some fuzziness. The HH model is deterministic, so in principle the input/output relation should be a $\delta$ function: spikes occur only when certain exact conditions are met. Of course we have blurred things a bit by working at finite time

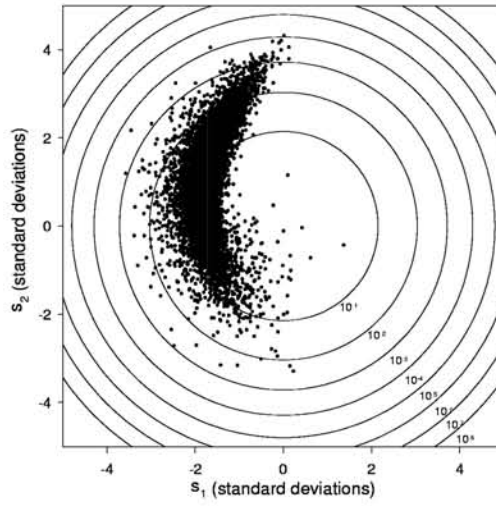

Figure 4: $10^4$ spike-conditional stimuli projected along the first 2 covariance modes. The circles represent the cumulative radial integral of the prior distribution from $\infty$; the ring marked $10^{-4}$, for example, encloses $1 - 10^{-4}$ of the prior.

resolution. Given that we work at finite $\Delta t$, spikes carry only a finite amount of information, and the quality of our 2D approximation can be judged by asking how much of this information is captured by this description.

As explained in [5], the arrival time of a single spike provides an information

$$I_{\text{one spike}} = \left\langle \frac{r(t)}{\bar{r}} \log_2 \left[ \frac{r(t)}{\bar{r}} \right] \right\rangle, \tag{10}$$

where $r(t)$ is the time dependent spike rate, $\bar{r}$ is the average spike rate, and $\langle \cdots \rangle$ denotes an average over time. With a deterministic model like HH, the rate $r(t)$ either is zero or corresponds to one spike occurring in one bin of size $\Delta t$, that is $r = 1/\Delta t$. The result is that $I_{\text{one spike}} = -\log_2(\bar{r}\Delta t)$.

On the other hand, if the probability of spiking really depends only on the stimulus dimensions $s_1$ and $s_2$, we can substitute

$$\frac{r(t)}{\bar{r}} \to \frac{P(s_1, s_2 | \text{spike@}t)}{P(s_1, s_2)}, \tag{11}$$

and use the ergodicity of the stimulus to replace time averages in Eq. (10). Then we find [3, 5]

$$I^{s_1, s_2}_{\text{one spike}} = \int d^2 s \, P(s_1, s_2 | \text{spike@}t) \log_2 \left[ \frac{P(s_1, s_2 | \text{spike@}t)}{P(s_1, s_2)} \right] \tag{12}$$

If our two dimensional approximation were exact we would find $I^{s_1, s_2}_{\text{one spike}} = I_{\text{one spike}}$; more generally we will find $I^{s_1, s_2}_{\text{one spike}} \leq I_{\text{one spike}}$, and the fraction of the information we capture measures the quality of the approximation. This fraction is plotted in fig. 5 as a function of time resolution. For comparison, we also show the information captured by considering only the stimulus projection along the STA.

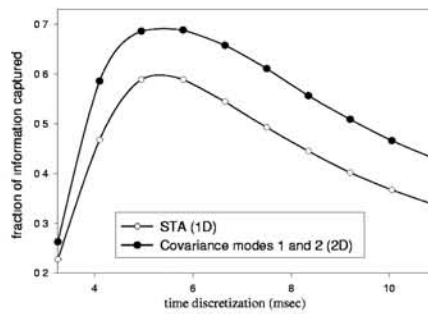

Figure 5: Fraction of spike timing information captured by STA (lower curve) and projection onto covariance modes 1 and 2 (upper curve).

# 7   Discussion

The simple, low-dimensional model described captures a substantial amount of information about spike timing for a HH neuron. The fraction is maximal near $\Delta t = 5.5$ msec, reaching nearly 70%. However, the absolute information captured saturates for both the 1D and 2D cases, at $\approx$ 3.5 and 5 bits respectively, for smaller $\Delta t$. Hence the information fraction captured plummets; recovering *precise* spike timing requires a more complex, higher dimensional representation of the stimulus.

Is this effect important, or is timing at this resolution too noisy for this extra complexity to matter in a real neuron? Stochastic HH simulations have suggested that, when realistic noise sources are taken into account, the timing of spikes in response to dynamic stimuli is reproducible to within 1–2 msec [6]. This suggests that such timing details may indeed be important.

Even in 2D, one can observe that the spike conditional distribution is curved (fig. 4); it is likely to curve along other dimensions as well. It may be possible to improve our approximation by considering the computation to take place on a low-dimensional but curved manifold, instead of a linear subspace. The curvature in Fig. 4 also implies that the computation in the HH model is not well approximated by an integrate and fire model, or a perceptron model limited to linear separations.

Characterizing the complexity of the computation is an important step toward understanding neural systems. How to quantify this complexity theoretically is an area for future work; here, we have made progress toward this goal by describing such computations in a compact way and then evaluating the completeness of the description using information. The techniques presented are applicable to more complex models, and of course to real neurons. How does the addition of more channels increase the complexity of the computation? Will this add more relevant dimensions or does the non-linearity change?

## Footnotes

[1]Note that the individual filters don't really have any meaning; what is meaningful is the projection operator that is formed by the whole set of these filters. Put another way, the individual filters specify both a $K$–dimensional subspace *and* a coordinate system on this subspace, but there is no reason to prefer one coordinate system over another.

[2] We have used the original parameters, with a sign change for voltages: $C = 1\mu F/cm^2$, $\bar{g}_K = 36m\mho/cm^2$, $\bar{g}_{Na} = 120m\mho/cm^2$, $\bar{g}_l = 0.3m\mho/cm^2$, $V_K = -12mV$, $V_{Na} = +115mV$, $V_l = +10.613mV$. We have taken our system to be a $\pi \times 30^2 \mu m^2$ patch of membrane.

# References

[1] A. Hodgkin and A. Huxley. *J. Physiol.*, 117, 1952.

[2] C. Koch. *Biophysics of computation.* New York: Oxford University Press, 1999.

[3] W. Bialek and R. de Ruyter van Steveninck. *Proc. R. Soc. Lond. B*, 234, 1988.

[4] F. Rieke, D. Warland, R. de Ruyter van Steveninck and W. Bialek. *Spikes: exploring the neural code.* Cambridge, MA: MIT Press, 1997.

[5] N. Brenner, S. Strong, R. Koberle, W. Bialek and R. de Ruyter van Steveninck. *Neural Comp.*, 12, 2000.

[6] E. Schneidman, R. Freedman and I. Segev. *Neural Comp.*, 10, 1998.
